# Identifying protein-protein interaction sites on a genome-wide scale

**Haidong Wang**[*]   **Eran Segal**[♮]   **Asa Ben-Hur**[†]   **Daphne Koller**[*]   **Douglas L. Brutlag**[‡]
[*] Computer Science Department, Stanford University, CA 94305
{haidong, koller}@cs.stanford.edu
[♮] Center for Studies in Physics and Biology, Rockefeller University, NY 10021
eran@cs.stanford.edu
[†] Department of Genome Sciences, University of Washington, WA 98195
asa@gs.washington.edu
[‡] Department of Biochemistry, Stanford University, CA 94305
brutlag@stanford.edu

## Abstract

Protein interactions typically arise from a physical interaction of one or more small sites on the surface of the two proteins. Identifying these sites is very important for drug and protein design. In this paper, we propose a computational method based on probabilistic relational model that attempts to address this task using high-throughput protein interaction data and a set of short sequence motifs. We learn the model using the EM algorithm, with a branch-and-bound algorithm as an approximate inference for the E-step. Our method searches for motifs whose presence in a pair of interacting proteins can explain their observed interaction. It also tries to determine which motif pairs have high affinity, and can therefore lead to an interaction. We show that our method is more accurate than others at predicting new protein-protein interactions. More importantly, by examining solved structures of protein complexes, we find that 2/3 of the predicted active motifs correspond to actual interaction sites.

## 1   Introduction

Many cellular functions are carried out through physical interactions between proteins. Discovering the protein interaction map can therefore help to better understand the workings of the cell. Indeed, there has been much work recently on developing high-throughput methods to produce a more complete map of protein-protein interactions [1, 2, 3].

Interactions between two proteins arise from physical interactions between small regions on the surface of the proteins [4] (see Fig. 2(b)). Finding interaction sites is an important task, which is of particular relevance to drug design. There is currently no high-throughput experimental method to achieve this goal, so computational methods are required. Existing methods either require solving a protein's 3D structure (e.g., [5]), and therefore are computationally very costly and not applicable on a genome-wide scale, or use known interaction sites as training data (e.g., [6]), which are relatively scarce and hence have poor coverage. Other work focuses on refining the highly noisy high-throughput interaction maps [7, 8, 9], or on assessing the confidence levels of the observed interactions [10].

In this paper, we propose a computational method for predicting protein interactions

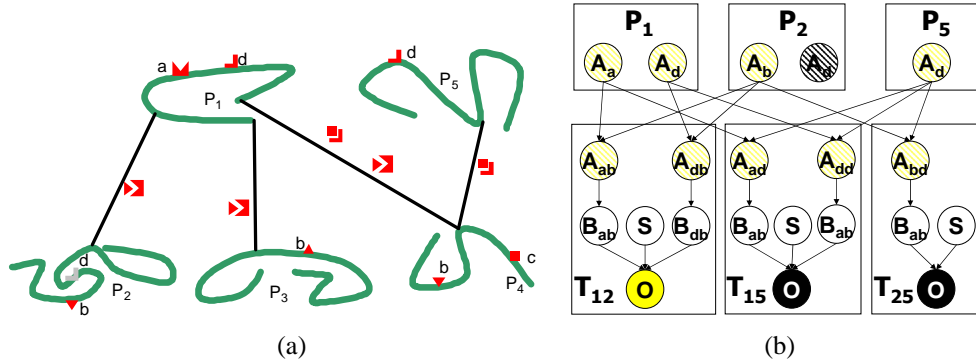

<div style="text-align:center">(a)           (b)</div>

Figure 1: (a) Simple illustration of our assumptions for protein-protein interactions. The small elements denote motif occurrences on proteins, with red denoting active and gray denoting inactive motifs. (b) A fragment of our probabilistic model, for the proteins $P_1, P_2, P_5$. We use yellow to denote an assignment of the value *true*, and black to denote the value *false*; full circles denote an assignment observed in the data, and patterned circles an assignment hypothesized by our algorithm. The dependencies involving inactive motif pairs were removed from the graph because they do not affect the rest of the model.

and the sites at which the interactions take place, which uses as input only high-throughput protein-protein interaction data and the protein sequences. In particular, our method assumes no knowledge of the 3D protein structure, or of the sites at which binding occurs.

Our approach is based on the assumption that interaction sites can be described using a limited repertoire of conserved *sequence motifs* [11]. This is a reasonable assumption since interaction sites are significantly more conserved than the rest of the protein surface [12]. Given a protein interaction map, our method tries to explain the observed interactions by identifying a set of sites of motif occurrence on every pair of interacting proteins through which the interaction is mediated. To understand the intuition behind our approach, consider the example of Fig. 1(a). Here, the interaction pattern of the protein $P_1$ can best be explained using the motif pair $a, b$, where $a$ appears in $P_1$ and $b$ in the proteins $P_2, P_3, P_4$ but not in $P_5$. By contrast, the motif pair $d, b$ is not as good an explanation, because $d$ also appears in $P_5$, which has a different interaction pattern. In general, our method aims to identify motif pairs that have high affinity, potentially leading to interaction between protein pairs that contain them.

However, a sequence motif might be used for a different purpose, and not give rise to an active binding site; it might also be buried inside the protein, and thus be inaccessible for interaction. Thus, the appearance of an appropriate motif does not always imply interaction. A key feature of our approach is that we allow each motif occurrence in a protein to be either *active* or *inactive*. Interactions are then induced only by the interactions of high-affinity *active* motifs in the two proteins. Thus, in our example, the motif $d$ in $p_2$ is inactive, and hence does not lead to an interaction between $p_2$ and $p_4$, despite the affinity between the motif pair $c, d$. We note that Deng *et al.* [8] proposed a somewhat related method for genome-wide analysis of protein interaction data, based on protein domains. However, their method is focused on predicting protein-protein interactions and not on revealing the site of interaction, and they do not allow for the possibility that some domains are inactive.

Our goal is thus to identify two components: the affinities between pairs of motifs, and the activity of the occurrences of motifs in different proteins. Our algorithm addresses this problem by using the framework of Bayesian networks [13] and probabilistic relational models [14], which allows us to handle the inherent noise in the protein interaction data and the uncertain relationship between interactions and motif pairs. We construct a model encoding our assumption that protein interactions are induced by the interactions of active motif pairs. We then use the EM algorithm [15], to fill in the details of the model, learning both the motif affinities and activities from the observed data of protein-protein interactions and protein motif occurrences. We address the computational complexity of the E-step in

these large, densely connected models by using an approximate inference procedure based on branch-and-bound.

We evaluated our model on protein-protein interactions in yeast and Prosite motifs [11]. As a basic performance measure, we evaluated the ability of our method to predict new protein-protein interactions, showing that it achieves better performance than several other models. In particular, our results validate our assumption that we can explain interactions via the interactions of active sequence motifs. More importantly, we analyze the ability of our method to discover the mechanism by which the interaction occurs. Finally, we examined co-crystallized protein pairs where the 3D structure of the interaction is known, so that we can determine the sites at which the interaction took place. We show that our active motifs are more likely to participate in interactions.

## 2 The Probabilistic Model

The basic entities in our probabilistic model are the proteins and the set of sequence motifs that can mediate protein interactions. Our model therefore contains a set of protein entities $\mathbf{P} = \{P_1, \ldots, P_n\}$, with the motifs that occur in them. Each protein $P$ is associated with the set of motifs that occur in it, denoted by $P.M$. As we discussed, a key premise of our approach is that a specific occurrence of a sequence motif may or may not be active. Thus, each motif occurrence $a \in P.M$ is associated with a binary-value variable $P.A_a$, which takes the value *true* if $A_a$ is active in protein $P$ and *false* otherwise. We structure the prior probability $P(P.A_a = true) = \min\{0.8, \frac{3+0.1 \cdot |P.M|}{|P.M|}\}$, to capture our intuition that the number of active motifs in a protein is roughly a constant fraction of the total number of motifs in the protein, but that even proteins with few motifs tend to have at least some number of active motifs.

A pair of active motifs in two proteins can potentially *bind* and induce an interaction between the corresponding proteins. Thus, in our model, a pair of proteins interact if each contains an active motif, and this pair of motifs bind to each other. The probability with which two motifs bind to each other is called their *affinity*. We encode this assumption by including in our model entities $T_{ij}$ corresponding to a pair of proteins $P_i, P_j$. For each pair of motifs $a \in P_i.M$ and $b \in P_j.M$, we introduce a variable $T_{ij}.A_{ab}$, which is a deterministic AND of the activity of these two motifs. Intuitively, this variable represents whether the pair of motifs can potentially interact. The probability with which two active motif occurrences bind is their affinity. We model the binding event between two motif occurrences using a variable $T_{ij}.B_{ab}$, and define: $P(T_{ij}.B_{ab} = true \mid T_{ij}.A_{ab} = true) = \theta_{ab}$ and $P(T_{ij}.B_{ab} = true \mid T_{ij}.A_{ab} = false) = 0$, where $\theta_{ab}$ is the affinity between motifs $a$ and $b$. This model reflects our assumption that two motif occurrences can bind only if they are both active, but their actual binding probability depends on their affinity. Note that this affinity is a feature of the motif pair and does not depend on the proteins in which they appear.

We must also account for interactions that are not explained by our set of motifs, whether because of false positives in the data, or because of inadequacies of our model or of our motif set. Thus, we add a *spurious binding* variable $T_{ij}.S$, for cases where an interaction between $P_i$ and $P_j$ exists, but cannot be explained well by our set of active motifs. The probability that a spurious binding occurs is given by $P(T_{ij}.S = true) = \theta_S$.

Finally, we observe an interaction between two proteins if and only if some form of binding occurs, whether by a motif pair or a spurious binding. Thus, we define a variable $T_{ij}.O$, which represents whether protein $i$ was observed to interact with protein $j$, to be a deterministic OR of all the binding variables $T_{ij}.S$ and $T_{ij}.B_{ab}$. Overall, $T_{ij}.O$ is a *noisy-OR* [13] of all motif pair variables $T_{ij}.A_{ab}$.

Note that our model accounts for both types of errors in the protein interaction data. False negatives (missing interactions) in the data are addressed through the fact that the presence of an active motif pair only implies that binding takes place with some probability. False positives (wrong interactions) in the data are addressed through the introduction of

the spurious interaction variables.

The full model defines a joint probability distribution over the entire set of attributes:

$$P(\mathbf{P.A}, \mathbf{T.A}, \mathbf{T.B}, \mathbf{T.S}, \mathbf{T.O}) = \prod_i \prod_{a \in P_i.M} P(P_i.A_a)$$
$$\prod_{ij} \left[ \begin{array}{l} \prod_{a \in P_i.M, b \in P_j.M} P(T_{ij}.A_{ab} \mid P_i.A_a, P_j.A_b)P(T_{ij}.B_{ab} \mid T_{ij}.A_{ab}) \\ P(T_{ij}.S)P(T_{ij}.O \mid T_{ij}.\mathbf{B}, T_{ij}.S) \end{array} \right]$$

where each of these conditional probability distributions is as specified above. We use $\Theta$ to denote the entire set of model parameters $\{\theta_{a,b}\}_{a,b} \cup \{\theta_S\}$. An instantiation of our probabilistic model is illustrated in Fig. 1(b).

## 3  Learning the Model

We now turn to the task of learning the model from the data. In a typical setting, we are given as input a protein interaction data set, specifying a set of proteins $\mathbf{P}$ and a set of observed interacting pairs $\mathbf{T.O}$. We are also given a set of potentially relevant motifs, and the occurrences of these motifs in the different proteins in $\mathbf{P}$. Thus, all the variables except for the $\mathbf{O}$ variables are hidden. Our learning task is thus twofold: we need to infer the values of the hidden variables, both the activity variables $\mathbf{P.A}$, $\mathbf{T.A}$, and the binding variables $\mathbf{T.B}$, $\mathbf{T.S}$; we also need to find a setting of the model parameters $\Theta$, which specify the motif affinities. We use a variant of the *EM* algorithm [15] to find both an assignment to the parameters $\Theta$, and an assignment to the motif variables $\mathbf{P.A}$, which is a local maximum of the likelihood function $P(\mathbf{T.O}, \mathbf{P.A} \mid \Theta)$. Note that, to maximize this objective, we search for a MAP assignment to the motif activity variables, but sum out over the other hidden variables. This design decision is reasonable in our setting, where determining motif activities is an important goal; it is a key assumption for our computational procedure.

As in most applications of EM, our main difficulty arises in the E-step, where we need to compute the distribution over the hidden variables given the settings of the observed variables and the current parameter settings. In our model, any two motif variables (both within the same protein and across different proteins) are correlated, as there exists a path of influence between them in the underlying Bayesian network (see Fig. 1(c)). These correlations make the task of computing the posterior distribution over the hidden variables intractable, and we must resort to an approximate computation. While we could apply a general purpose approximate inference algorithm such as loopy belief propagation [16], such methods may not converge in densely connected model such as this one, and there are few guarantees on the quality of the results even if they do converge. Fortunately, our model turns out to have additional structure that we can exploit. We now describe an approximate inference algorithm that is tailored to our model, and is guaranteed to converge to a (strong) local maximum.

Our first observation is that the only variables that correlate the different protein pairs $T_{ij}$ are the motif variables $\mathbf{P.A}$. Given an assignment to these activity variables, the network decomposes into a set of independent subnetworks, one for each protein pair. Based on this observation, we divide our computation of the E-step into two parts. In the first, we find an assignment to the motif variables in each protein, $\mathbf{P.A}$; in the second, we compute the posterior probability over the binding motif pair variables $\mathbf{T.B}$, $\mathbf{T.S}$, given the assignment to the motif variables.

We begin by describing the second phase. We observe that, as all the motif pair variables, $\mathbf{T.A}$, are fully determined by the motif variables, the only variables left to reason about are the binding variables $\mathbf{T.B}$ and $\mathbf{T.S}$. The variables for any pair $T_{ij}$ are independent of the rest of the model given the instantiation to $\mathbf{T.A}$ and the interaction evidence. That fact, combined with the noisy-OR form of the interaction, allows us to compute the posterior probability required in the E-step exactly and efficiently. Specifically, the computation for the variables associated with a particular protein pair $T_{ij}$ is as follows, where we omit the common prefix $T_{ij}$ to simplify notation. If $A_{ab} = $ *false*, then

$P(B_{ab} = \textit{true} \mid A_{ab} = \textit{false}, O, \Theta) = 0$. Otherwise, if $A_{ab} = \textit{true}$, then

$$P(B_{ab} = \textit{true} \mid \mathbf{A}, O, \Theta) \quad = \quad \frac{P(B_{ab} \mid \mathbf{A}, \Theta) P(O \mid B_{ab} = \textit{true}, \mathbf{A}, \Theta)}{P(O \mid \mathbf{A}, \Theta)}.$$

The first term in the numerator is simply the motif affinity $\theta_{ab}$; the second term is 1 if $O = \textit{true}$ and 0 otherwise. The numerator can easily be computed as $P(O \mid \mathbf{A}, \Theta) = 1 - (1 - \theta_S) \prod_{A_{a,b}=\textit{true}} (1 - \theta_{ab})$. The computation for $P(S)$ is very similar.

We now turn to the first phase, of finding a setting to all of the motif variables. Unfortunately, as we discussed, the model is highly interconnected, and a finding an optimal joint setting to all of these variables $\mathbf{P}.\mathbf{A}$ is intractable. We thus approximate finding this joint assignment using a method that exploits our specific structure. Our method iterates over proteins, finding in each iteration the optimal assignment to the motif variables of each protein given the current assignment to the motif activities in the remaining proteins. The process repeats, iterating over proteins, until convergence.

As we discussed, the likelihood of each assignment to $P_i.\mathbf{A}$ can be easily computed using the method described above. However, the computation for each protein is still exponential in the number of motifs it contains, which can be large (e.g., 15). However, in our specific model, we can apply the following branch-and-bound algorithm (similar to an approach proposed by Henrion [17] for BN2O networks) to find the globally optimal assignment to the motif variables of each protein. The idea is that we search over the space of possible assignments $P_i.\mathbf{A}$ for one that maximizes the objective we wish to maximize. We can show that if making a motif active relative to one assignment does not improve the objective, it will also not improve the objective relative to a large set of other assignments.

More precisely, let $f(P_i.\mathbf{A}) = P(P_i.\mathbf{A}, \mathbf{P}_{-i}.\mathbf{A} \mid O, \theta)$ denote the objective we wish to maximize, where $\mathbf{P}_{-i}.\mathbf{A}$ is the fixed assignment to motif variables in all proteins except $P_i$. Let $P_i.\mathbf{A}_{-a}$ denote the assignment to all the motif variables in $P_i$ except for $A_a$. We compute the ratio of $f$ after we switch $P_i.A_a$ from $\textit{false}$ to $\textit{true}$. Let $h_a(P_j) = \prod_{P_j.A_b=\textit{true}} (1 - \theta_{ab})$ denote the probability that motif $a$ does not bind with any active motif in $P_j$. We can now compute:

$$\Delta_a(P_i.\mathbf{A}_{-a}) = \frac{f(P_i.A_a = \textit{true}, P_i.\mathbf{A}_{-a})}{f(P_i.A_a = \textit{false}, P_i.\mathbf{A}_{-a})} = \frac{g}{1-g}$$

$$\cdot \prod_{\substack{1 \le j \le n \\ T_{ij}.O=\textit{false}}} h_a(P_j) \cdot \prod_{\substack{1 \le j \le n \\ T_{ij}.O=\textit{true}}} \frac{1 - (1-\theta_S)h_a(P_j) \prod_{a \ne b, P_i.A_b=\textit{true}} h_b(P_j)}{1 - (1-\theta_S) \prod_{a \ne b, P_i.A_b=\textit{true}} h_b(P_j)} \quad (1)$$

where $g$ is the prior probability for a motif in protein $P_i$ to be active.

Now, consider a different point in the search, where our current motif activity assignment is $P_i.\mathbf{A}'_{-a}$, which has all the active motifs in $P_i.\mathbf{A}_{-a}$ and some additional ones. The first two terms in the product of Eq. (1) are the same for $\Delta_a(P_i.\mathbf{A}_{-a})$ and $\Delta_a(P_i.\mathbf{A}'_{-a})$. For the final term (the large fraction), one can show using some algebraic manipulation that this term in $\Delta_a(P_i.\mathbf{A}'_{-a})$ is lower than that for $\Delta_a(P_i.\mathbf{A}_{-a})$. We conclude that $\Delta_a(P_i.\mathbf{A}_{-a}) \ge \Delta_a(P_i.\mathbf{A}'_{-a})$, and hence that:

$$\frac{f(P_i.A_a = \textit{true}, P_i.\mathbf{A}_{-a})}{f(P_i.A_a = \textit{false}, P_i.\mathbf{A}_{-a})} \le 1 \implies \frac{f(P_i.A_a = \textit{true}, P_i.\mathbf{A}'_{-a})}{f(P_i.A_a = \textit{false}, P_i.\mathbf{A}'_{-a})} \le 1.$$

It follows that, if switching motif $a$ from inactive to active relative to $P_i.\mathbf{A}$ decreases $f$, it will also decrease $f$ if we have some additional active motifs.

We can exploit this property in a branch-and-bound algorithm in order to find the globally optimal assignment $P_i.\mathbf{A}$. Our algorithm keeps a set $V$ of viable candidates for motif assignments. For presentation, we encode assignments via the set of active motifs they contain. Initially, $V$ contains only the empty assignment $\{\}$. We start out by considering

motif assignments with a single active motif. We put such an assignment $\{a\}$ in $V$ if its $f$-score is higher than $f(\{\})$. Now, we consider assignments $\{a, b\}$ that have two active motifs. We consider $\{a, b\}$ only if both $\{a\}$ and $\{b\}$ are in $V$. If so, we evaluate its $f$-score, and add it to $V$ if this score is greater than that of $\{a\}$ and $\{b\}$. Otherwise, we throw it away. We continue this process for all assignments of size $k$: For each assignment with active motif set $S$, we test whether $S - \{a\} \in V$ for all $a \in S$; if we compare $f(S)$ to each $f(S - \{a\})$, and add it if it dominates all of them. The algorithm terminates when, from some $k$, no assignment of size $k$ is saved.

To understand the intuition behind this pruning procedure, consider a candidate assignment $\{a, b, c, d\}$, and assume that $\{a, b, c\} \in V$, but $\{b, c, d\} \notin V$. In this case, we must have that $\{b, c\} \in V$, but adding $d$ to that assignment reduces the $f$-score. In this case, as shown by our analysis, adding $d$ to the superset $\{a, b, c\}$ would also reduce the $f$-score.

This algorithm is still exponential in worst case. However, in our setting, a protein with many motifs has a low prior probability that each of them is active. Hence, adding new motifs is less likely to increase the $f$-score, and the algorithm tends to terminate quickly. As we show in Section 4, this algorithm significantly reduces the cost of our procedure.

Our E-step finds an assignment to $\mathbf{P}.\mathbf{A}$ which is a strong local optimum of the objective function $\max P(\mathbf{P}.\mathbf{A} \mid \mathbf{T}.\mathbf{O}, \Theta)$: The assignment has higher probability than any assignment that changes any of the motif variables for any single protein. For that assignment, our algorithm also computes the distribution over all of the binding variables, as described above. Using this completion, we can now easily compute the (expected) sufficient statistics for the different parameters in the model. As each of these parameters is a simple binomial distribution, the maximum likelihood estimation in the M-step is entirely standard; we omit details.

## 4 Results

We evaluated our model on reliable *S. cerevisiae* protein interactions data from MIPS [2] and DIP [3] databases. As for non-interaction data, we randomly picked pairs of proteins that have no common function and cellular location. This results in a dataset of 2275 proteins, 4838 interactions ($T_{ij}.O = true$), and 9037 non-interactions ($T_{ij}.O = false$). We used sequence motifs from the Prosite database [11] resulting in a dataset of 516 different motifs with an average of 7.1 motif occurrences per protein. If a motif pair doesn't appear between any pair of interacting proteins, we initialize its affinity to be 0 to maximize the joint likelihood. Its affinity will stay at 0 during the EM iterations and thus simplify our model structure. We set the initial affinity for the remaining 8475 motif pairs to 0.03.

We train our model with motifs initialized to be either all active ($\mathbf{P}.\mathbf{A} = true$) or all inactive ($\mathbf{P}.\mathbf{A} = false$). We get similar results with these two different initializations, indicating the robustness of our algorithm. Below we only report the results based on all motifs initialized to be active. Our branch-and-bound algorithm is able to significantly reduce the number of motif activity assignments that need to be evaluated. For a protein with 15 motifs, the number of assignments evaluated is reduced from $2^{15} = 32768$ in exhaustive search to 802 using our algorithm. Since majority of the computation is spent on finding the activity assignments, this resulted in a 40 fold reduction in running time.

**Predicting protein-protein interactions.** We test our model by evaluating its performance in predicting interactions. We test this performance using 5-fold cross validation on the set of interacting and non-interacting protein pairs. In each fold, we train a model and predict $P(T_{ij}.O) = true$ for pairs $P_i, P_j$ in the held-out interactions.

Many motif pairs are over-represented in interacting proteins. We thus compare our method to a baseline method that ranks pairs of proteins on the basis of the maximum enrichment of over-represented motif pairs (see [18] for details). We also compare it to a model where all motifs are set to be active; this is analogous to the method of Deng *et al.* [8]. For completeness, we compare the two variants of the model using data on the domain (Pfam and ProDom [19]) content of the proteins as well as the Prosite motif content.

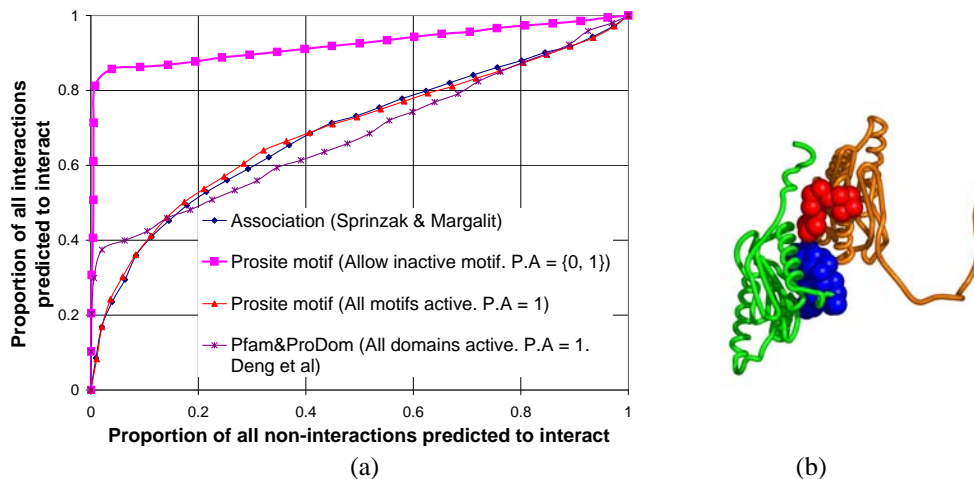

(a)                                                                (b)

Figure 2: (a) ROC curve for different methods. The X-axis is the proportion of all non-interacting protein pairs in the training data predicted to interact. Y-axis is the proportion of all interacting protein pairs in the training data predicted to interact. Points are generated using different cutoff probabilties. A larger area under the curve indicates better prediction. Our method (square marker) outperforms all other methods. (b) Two protein chains that form a part of the 1ryp complex in PDB, interacting at the site of two short sequence motifs.

The ROC curves in Fig. 2(a) show that our method outperforms the other methods, and that the additional degree of freedom of allowing motifs to be inactive is essential. These results validate our modeling assumptions; they also show that our method can be used to suggest new interactions and to assign confidence levels on observed interactions, which is much needed in view of the inaccuracies and large fraction of missing interactions in current interaction databases.

**Evaluating predicted active motifs.** A key feature of our approach is its ability to detect pairs of interacting motifs. We evaluate these predictions against the data from Protein Data Bank (PDB) [20], which contains some solved structures of interacting proteins Fig. 2(b). While the PDB data is scarce, it provides the ultimate evaluation of our predicted active motifs. We extracted all structures from PDB that have at least two co-crystallized chains, and whose chains are nearly identical to yeast proteins. From the residues that are in contact between two chains (distance $< 5$ Angströms), we infer which protein motifs participate in interactions. Among our training data, 105 proteins have co-crystallized structure in PDB. On these proteins, our data contained a total of 620 motif occurrences, of which 386 are predicted to be active. Among those motifs predicted to be active, 257 of them (66.6%) are interacting in PDB. Among the 234 motifs predicted to be inactive, only 120 of them (51.3%) are interacting. The chi-square p-value is $10^{-4}$. On the residue level, our predicted active motifs consist of 3736 amino acids, and 1388 of them (37.2%) are interacting. In comparison, our predicted inactive motifs consist of 3506 amino acids, and only 588 of them (16.0%) are interacting. This significant enrichment provides support for the ability of our method to detect motifs that participate in interactions. In fact, the set of interactions in PDB is only a subset of the interactions those proteins participate in. Therefore, the actual rate of false positive active motifs is likely to be lower than we report here.

## 5   Discussion and Conclusions

In this paper, we presented a probabilistic model which explicitly encodes elements in the protein sequence that mediate protein-protein interactions. By using a variant of the EM al-

gorithm and a branch-and-bound algorithm for the E-step, we make the learning procedure tractable. Our result shows that our method successfully uncovers motif activities and binding affinities, and uses them to predict both protein interactions and specific binding sites. The ability of our model to predict structural elements, without a full structure analysis, provides support for the viability of our approach.

Our use of a probabilistic model provides us with a general framework to incorporate different types of data into our model, allowing it to be extended in varies ways. First, we can incorporate additional signals for protein interactions, such as gene expression data (as in [9]), cellular location, or even annotations from the literature (as in [7]). We can also integrate protein interaction data across multiple species; for example, we might try to use the yeast interaction data to provide more accurate predictions for the protein-protein interactions in fly [10].

# References

[1] P. Uetz, *et al.* A comprehensive analysis of protein-protein interactions in saccharomyces cerevisiae. *Nature*, 403(6770):623–7, 2000. 0028-0836 Journal Article.

[2] H. W. Mewes, *et al.* Mips: a database for genomes and protein sequences. *Nucleic Acids Res*, 2002.

[3] I. Xenarios, *et al.* Dip ; the database of interacting proteins: a research tool for studying cellular networks of protein interactions. *Nucleic Acids Research*, 30(1):303–305, 2002. (c) 2002 Inst. For Sci. Info.

[4] P. Chakrabarti and J. Janin. Dissecting protein protein recognition sites. *PROTEINS: Structure, Function, and Genetics*, 47:334–343, 2002.

[5] J. J. Gray, *et al.* Protein protein docking with simultaneous optimization of rigid-body displacement and side-chain conformations. *Journal of Molecular Biology*, 331:281–299, 2003.

[6] Y. Ofran and B. Rost. Predicted protein-protein interaction sites from local sequence information. *FEBS Lett.*, 544(1-3):236–239, 2003.

[7] R. Jansen, *et al.* A bayesian networks approach for predicting protein-protein interactions from genomic data. *Science*, 302:449–53, 2003.

[8] M. Deng, S. Mehta, F. Sun, and T. Chen. Inferring domain-domain interactions from protein-protein interactions. *Genome Res*, 12(10):1540–8, 2002. 22253763 1088-9051 Journal Article.

[9] E. Segal, H. Wang, and D. Koller. Discovering molecular pathways from protein interaction and gene expression data. *Bioinformatics*, 19 Suppl 1:I264–I272, 2003. 1367-4803 Journal Article.

[10] L. Giot, *et al.* A protein interaction map of drosophila melanogaster. *Science*, 302(5651):1727–36, 2003.

[11] L. Falquet, *et al.* The PROSITE database, its status in 2002. *Nucliec Acids Research*, 30:235–238, 2002.

[12] D. R. Caffrey, *et al.* Are protein protein interfaces more conserved in sequence than the rest of the protein surface? *Protein Science*, 13:190–202, 2003.

[13] J. Pearl. *Probabilistic Reasoning in Intelligent Systems*. Morgan Kaufmann, 1988.

[14] D. Koller and A. Pfeffer. Probabilistic frame-based systems. In *Proc. AAAI*, pages 580–587, 1998.

[15] A.P. Dempster, N.M. Laird, and D.B. Rubin. Maximum likelihood from incomplete data via the em algorithm. *J. Roy. Stat. Soc.*, B(39):1–39, 1977.

[16] J. S. Yedidia, W. T. Freeman, and Y. Weiss. Generalized belief propagation. In *NIPS*, pages 689–695, 2000.

[17] M. Henrion. Search-based methods to bound diagnostic probabilities in very large belief nets. In *Uncertainty in Artificial Intelligence*, pages 142–150, 1991.

[18] E. Sprinzak and H. Margalit. Correlated sequence-signatures as markers of protein-protein interaction. *Journal of Molecular Biology*, 311:681–692, 2001.

[19] R. Apweiler, *et al.* The interpro database, an integrated documentation resource for protein families, domains and functional sites. *Nucleic Acids Res*, 29(1):37–40, 2001. 1362-4962 Journal Article.

[20] H.M. Berman, *et al.* The protein data bank. *Nucleic Acids Research*, 28:235–242, 2000.
